# Submodular-Bregman and the Lovász-Bregman Divergences with Applications

**Rishabh Iyer**
Department of Electrical Engineering
University of Washington
rkiyer@u.washington.edu

**Jeff Bilmes**
Department of Electrical Engineering
University of Washington
bilmes@uw.edu

## Abstract

We introduce a class of discrete divergences on sets (equivalently binary vectors) that we call the submodular-Bregman divergences. We consider two kinds, defined either from tight modular upper or tight modular lower bounds of a submodular function. We show that the properties of these divergences are analogous to the (standard continuous) Bregman divergence. We demonstrate how they generalize many useful divergences, including the weighted Hamming distance, squared weighted Hamming, weighted precision, recall, conditional mutual information, and a generalized KL-divergence on sets. We also show that the generalized Bregman divergence on the Lovász extension of a submodular function, which we call the Lovász-Bregman divergence, is a continuous extension of a submodular Bregman divergence. We point out a number of applications, and in particular show that a proximal algorithm defined through the submodular Bregman divergence provides a framework for many mirror-descent style algorithms related to submodular function optimization. We also show that a generalization of the k-means algorithm using the Lovász Bregman divergence is natural in clustering scenarios where ordering is important. A unique property of this algorithm is that computing the mean ordering is extremely efficient unlike other order based distance measures.

## 1 Introduction

The Bregman divergence first appeared in the context of relaxation techniques in convex programming ([4]), and has found numerous applications as a general framework in clustering ([2]), proximal minimization ([5]) and online learning ([27]). Many of these applications are due to the nice properties of the Bregman divergence, and the fact that they are parameterized by a single convex function. They also generalize a large class of divergences on vectors. Recently Bregman divergences have also been defined between matrices ([26, 6]) and between functions ([8]).

In this paper we define a class of divergences between sets, where each divergence is parameterized by a submodular function. This can alternatively and equivalently be seen as a divergence between binary vectors in the same way that submodular functions are special cases of pseudo-Boolean functions [3]. We call this the class of *submodular Bregman divergences* (or just *submodular Bregman*). We show an interesting mathematical property of the submodular Bregman, namely that they can be defined based on either a tight modular (linear) upper bound or *alternatively* a tight modular lower bound, unlike the traditional (continuous) Bregman definable only via a tight linear lower bound.

Let $V$ refer to a finite ground set $\{1, 2, \ldots, |V|\}$. A set function $f : 2^V \to \mathbb{R}$ is submodular if $\forall S, T \subseteq V, f(S) + f(T) \geq f(S \cup T) + f(S \cap T)$. Submodular functions have attractive properties that make their exact or approximate optimization efficient and often practical. Submodularity can be seen as a discrete counterpart to convexity and concavity ([20]) and often the problems are closely related ([1]). Indeed, as we shall see in this paper, the connections between submodularity

and convexity and concavity will help us formulate certain discrete divergences that are analogous to the Bregman divergence. We in fact show a direct connection between a submodular Bregman and a generalized Bregman divergence defined through the Lovász extension. Further background on submodular functions may be found in the text [9].

An outline of the paper follows. We first define the different types of submodular Bregman in Section 2. We also define the Lovász Bregman divergence, and show its relation to a version of the submodular Bregman. Then in Section 3, we prove a number of properties of the submodular Bregman and show how they are related to the Bregman divergence. Finally in Section 4, we show how the submodular Bregman can be used in applications in machine learning. In particular, we show how the proximal framework of the submodular Bregman generalizes a number of mirror-descent style approximate submodular optimization algorithms. We also consider generalizations of the $k$-means algorithm using the Lovász Bregman divergence, and show how they can be used in clustering applications where ordering or ranking is important.

## 2 The Bregman and Submodular Bregman divergences

Notation: We use $\phi$ to refer to a convex function, $f$ to refer to a submodular function, and $\hat{f}$ as $f$'s Lovász extension. Lowercase characters $x, y$ will refer to continuous vectors, while upper case characters $X, Y, S$ will refer to sets. We will also refer to the characteristic vectors of a set $X$ as $1_X \in \{0, 1\}^V$. Note that the characteristic vector of a set $X$, $1_X$ is such that $1_X(j) = I(j \in X)$, where $I(\cdot)$ is the standard indicator function. We will refer to the ground set as $V$, and the cardinality of the ground set as $n = |V|$. A divergence on vectors and sets is formally defined as follows: Given a domain of vectors or sets $\mathbb{S}$ (and if sets, $\mathbb{S} = $ a lattice of sets $\mathcal{L}$, where $\mathcal{L}$ is a lattice if $\forall X, Y \in \mathcal{L}, X \cup Y, X \cap Y \in \mathcal{L}$), a function $d : \mathbb{S} \times \mathbb{S} \to \mathbb{R}_+$ is called a *divergence* if $\forall x, y \in \mathbb{S}$, $d(x, y) \geq 0$ and $\forall x \in \mathbb{S}, d(x, x) = 0$. For simplicity, we consider mostly the Boolean lattice $\mathcal{L} = 2^V$ but generalizations are possible as well [9].

### 2.1 Bregman and Generalized Bregman divergences

Recall the definition of the Bregman divergence: $d_\phi : \mathbb{S} \times \mathbb{S} \to \mathbb{R}_+$ as:

$$d_\phi(x, y) = \phi(x) - \phi(y) - \langle \nabla \phi(y), x - y \rangle. \tag{1}$$

For non-differentiable convex functions we can extend equation (1) to define the generalized Bregman divergence [13, 18]. Define a subgradient map $\mathcal{H}_\phi$, which for every vector $y$, gives a subgradient $\mathcal{H}_\phi(y) = h_y \in \partial \phi(y)$ [13], where $\partial \phi(y)$ is the subdifferential of $\phi$ at $y$.

$$d_\phi^{\mathcal{H}_\phi}(x, y) = \phi(x) - \phi(y) - \langle \mathcal{H}_\phi(y), x - y \rangle, \forall x, y \in \mathbb{S}. \tag{2}$$

When $\phi$ is differentiable, then $\partial \phi(x) = \{\nabla \phi(x)\}$ and $\mathcal{H}_\phi(y) = \nabla \phi(y)$. More generally, there may be multiple distinct subgradients in the subdifferential, hence the generalized Bregman divergence is parameterized both by $\phi$ and the subgradient-map $\mathcal{H}_\phi$. The generalized Bregman divergences have also been defined in terms of "extreme" subgradients [25, 18].

$$d_\phi^\sharp(x, y) = \phi(x) - \phi(y) - \sigma_{\partial \phi(y)}(x - y) \quad \text{and} \quad d_\phi^\flat(x, y) = \phi(x) - \phi(y) + \sigma_{\partial \phi(y)}(y - x), \tag{3}$$

where, for a convex set $C$, $\sigma_C(.) \triangleq \max_{x \in C} \langle ., x \rangle$. Clearly, we then have: $d_\phi^\sharp(x, y) \leq d_\phi^{\mathcal{H}_\phi}(x, y) \leq d_\phi^\flat(x, y), \forall \mathcal{H}_\phi$ which justifies their being called the extreme generalized Bregman divergences [13].

### 2.2 The Submodular Bregman divergences

In a similar spirit, we define a submodular Bregman divergence parameterized by a submodular function and defined as the difference between the function and its modular (sometimes called linear) bounds. Surprisingly, any submodular function has both a tight upper and lower modular bound ([15]), unlike strict convexity where only a tight first-order lower bound exists. Hence, we define two distinct forms of submodular Bregman parameterized by a submodular function and in terms of either its tight upper or tight lower bounds.

### 2.2.1 Lower bound form of the Submodular Bregman

Given a submodular function $f$, the submodular polymatroid $\mathcal{P}_f$, the corresponding base polytope $\mathcal{B}_f$ and the subdifferential $\partial_f(Y)$ (at a set $Y$) for a submodular function $f$ [9] are respectively:

$$\mathcal{P}_f = \{x : x(S) \leq f(S), \forall S \subseteq V\}, \qquad \mathcal{B}_f = \mathcal{P}_f \cap \{x : x(V) = f(V)\}, \text{ and} \qquad (4)$$

$$\partial_f(Y) = \{y \in \mathbb{R}^V : \forall X \subseteq V, f(Y) - y(Y) \leq f(X) - y(X)\}. \qquad (5)$$

Note that here $y(S) = \sum_{j \in S} y(j)$ is a modular function. In a manner similar to the generalized Bregman divergence ([13]), we define a discrete subgradient map for a submodular function $\mathcal{H}_f$, which for every set $Y$, picks a subgradient $\mathcal{H}_f(Y) = h_Y \in \partial_f(Y)$. Then, given a submodular function $f$ and a subgradient-map $\mathcal{H}_f$, the generalized lower bound submodular Bregman, which we shall henceforth call $d_f^{\mathcal{H}_f}$, is defined as:

$$d_f^{\mathcal{H}_f}(X, Y) = f(X) - f(Y) - h_Y(X) + h_Y(Y) = f(X) - f(Y) - \langle \mathcal{H}_f(Y), 1_X - 1_Y \rangle. \quad (6)$$

We remark here that similar to the definition of the generalized Bregman divergence, this submodular Bregman is parameterized both by the submodular function $f$ and the subgradient map $\mathcal{H}_f$.

The subdifferential corresponding to a submodular function is an unbounded polyhedron [9], with an uncountable number of possible subgradients. Its extreme points, however, are easy to find and characterize using the greedy algorithm [7]. Thus, we define a subclass of $d_f^{\mathcal{H}_f}$ with $\mathcal{H}_f$ chosen so that it picks an extreme points of $\partial_f(Y)$, which we will call the permutation based lower bound submodular Bregman, henceforth referred to with $d_f^\Sigma$. The extreme points of $\partial_f(Y)$ can be obtained via a greedy algorithm ([7, 9]) as follows: Let $\sigma$ be a permutation of $V$ and define $S_i = \{\sigma(1), \sigma(2), \ldots, \sigma(i)\}$ as its corresponding chain. We define $\Sigma_Y$ as the set of permutations $\sigma_Y$ such that their corresponding chains contain $Y$, meaning $S_{|Y|} = Y$. Then we can define a subgradient $h_{Y, \sigma_Y}$ (which is an extreme point of $\partial_f(Y)$) where:

$$\forall \sigma_Y \in \Sigma_Y, \;\; h_{Y, \sigma_Y}(\sigma_Y(i)) = \begin{cases} f(S_1) & \text{if } i = 1 \\ f(S_i) - f(S_{i-1}) & \text{otherwise} \end{cases}. \qquad (7)$$

In the above, $h_{Y, \sigma_Y}(Y) = f(Y)$. Hence define $\mathcal{H}_f^\Sigma$ as a subgradient map which picks a subgradient $h_{Y, \sigma_Y}$, for some $\Sigma(Y) = \sigma_Y \in \Sigma_Y$. Here we treat $\Sigma$ as a permutation operator which, for a given set $Y$, produces a permutation $\sigma_Y \in \Sigma_Y$. Hence we can rewrite Eqn. (6), with the above subgradient as

$$d_f^\Sigma(X, Y) = f(X) - h_{Y, \sigma_Y}(X) = f(X) - \langle \mathcal{H}_f^\Sigma(Y), 1_X \rangle. \qquad (8)$$

As can readily be seen, the $d_f^\Sigma$ are special cases of the $d_f^{\mathcal{H}_f}$. Similar to the extreme generalized Bregman divergence above, we can define forms of the "extreme" lower bound submodular Bregman divergences $d_f^\sharp(X, Y)$ and $d_f^\flat(X, Y)$. Since in the case of a submodular function $\partial_f(Y)$ is an unbounded polyhedron, we restrict $C = \partial_f(Y) \cap \mathcal{P}_f$, and define: $d_f^\sharp(X, Y) = f(X) - f(Y) - \sigma_C(1_X - 1_Y)$ and $d_f^\flat(X, Y) = f(X) - f(Y) + \sigma_C(1_Y - 1_X)$ The extreme lower bound submodular Bregman have very nice forms as shown in the theorem below:

**Theorem 2.1.** *For every* $h_Y \in \partial_f(Y) \cap \mathcal{P}_f, d_f^\sharp(X, Y) \leq d_f^{\mathcal{H}_f}(X, Y) \leq d_f^\flat(X, Y)$. *Similarly for every permutation map* $\Sigma$, $d_f^\sharp(X, Y) \leq d_f^\Sigma(X, Y) \leq d_f^\flat(X, Y)$. *Further* $d_f^\sharp(X, Y) = f(X) + f(Y) - f(X \cap Y) - f(X \cup Y)$. *Similarly* $d_f^\flat(X, Y) = f(X) - f(Y) - f(Y \backslash X) - f(V) + f(V \backslash X \backslash Y)$

The above theorem gives bounds for $d_f^{\mathcal{H}_f}$ and $d_f^\Sigma$. Further we see that $d_f^\sharp$ is exactly the divergence which defines the submodularity of $f$. Also notice that this is unlike the generalized Bregman divergences, where the "extreme" forms may not be easy to obtain in general [13].

### 2.2.2 The upper bound submodular Bregman

For submodular $f$, [23] established properties of submodular function using which we can define the following divergences (which we call here the Nemhauser divergences):

$$d_{\sharp}^{f}(X,Y) \triangleq f(X) - \sum_{j \in X \setminus Y} f(j|X - \{j\}) + \sum_{j \in Y \setminus X} f(j|X \cap Y) - f(Y) \qquad (9)$$

$$d_{\natural}^{f}(X,Y) \triangleq f(X) - \sum_{j \in X \setminus Y} f(j|X \cup Y - \{j\}) + \sum_{j \in Y \setminus X} f(j|X) - f(Y), \qquad (10)$$

where $f(j|X) \triangleq f(X \cup j) - f(X)$. Similar to the approach in ([15]), we can relax the Nemhauser divergences to obtain three modular upper bound submodular Bregmans as:

$$d_{1}^{f}(X,Y) \triangleq f(X) - \sum_{j \in X \setminus Y} f(j|X - \{j\}) + \sum_{j \in Y \setminus X} f(j|\emptyset) - f(Y), \qquad (11)$$

$$d_{2}^{f}(X,Y) \triangleq f(X) - \sum_{j \in X \setminus Y} f(j|V - \{j\}) + \sum_{j \in Y \setminus X} f(j|X) - f(Y). \qquad (12)$$

$$d_{3}^{f}(X,Y) \triangleq f(X) - \sum_{j \in X \setminus Y} f(j|V - \{j\}) + \sum_{j \in Y \setminus X} f(j|\emptyset) - f(Y). \qquad (13)$$

We call these the Nemhauser based upper-bound submodular Bregmans of, respectively, type-I, II and III. Henceforth, we shall refer to them as $d_{1}^{f}$, $d_{2}^{f}$ and $d_{3}^{f}$ and when referring to them collectively, we will use $d_{1:3}^{f}$. The Nemhauser divergences are analogous to the extreme divergences of the generalized Bregman divergences since they bound the Nemhauser based submodular Bregmans. Its not hard to observe that for a submodular function $f$, $d_{3}^{f}(X,Y) \geq d_{1}^{f}(X,Y) \geq d_{\sharp}^{f}(X,Y)$. Similarly $d_{3}^{f}(X,Y) \geq d_{2}^{f}(X,Y) \geq d_{\natural}^{f}(X,Y)$

Similar to the generalized lower bound submodular Bregman $d_{f}^{\mathcal{H}_{f}}$, we define a generalized upper bound submodular Bregman divergence $d_{\mathcal{G}^{f}}^{f}$ in terms of any supergradient of $f$. Interestingly for a submodular function, we can define a superdifferential $\partial^{f}(X)$ at $X$ as follows:

$$\partial^{f}(X) = \{x \in \mathbb{R}^{V} : \forall Y \subseteq V, f(X) - x(X) \geq f(Y) - x(Y)\}. \qquad (14)$$

Given a supergradient at $X$, $g_X \in \partial^{f}(X)$, we can define a divergence $d_{\mathcal{G}^{f}}^{f}$, as:

$$d_{\mathcal{G}^{f}}^{f}(X,Y) = f(X) - f(Y) - g_X(X) - g_X(Y) = f(X) - f(Y) - \langle g_X, 1_X - 1_Y \rangle \qquad (15)$$

Similar to the subgradient map, we can define $\mathcal{G}^{f}$ as the supergradient map, which picks a supergradient from $\mathcal{G}^{f}(X) = g_X \in \partial^{f}(X)$. In fact, it can be shown that all three forms of $d_{1:3}^{f}$ are actually special cases of $d_{\mathcal{G}^{f}}^{f}$, in that they form specific supergradient maps. Define three supergradients $g_X^1, g_X^2$ and $g_X^3$ (with the corresponding maps $\mathcal{G}_1^f, \mathcal{G}_2^f$ and $\mathcal{G}_3^f$) such that: $g_X^1(j) = f(j|X - \{j\})$ and $g_X^2(j) = g_X^3(j) = f(j|V - \{j\})$ for $j \in X$. Similarly let $g_X^1(j) = g_X^3(j) = f(j|\emptyset)$ and $g_X^2(j) = f(j|X)$ for $j \notin X$. Then it can be shown [12] that $g_X^1, g_X^2, g_X^3 \in \partial^{f}(X)$, and correspondingly $d_1^f$, $d_2^f$ and $d_3^f$ are special cases of $d_{\mathcal{G}^{f}}^{f}$.

$d_{\mathcal{G}^{f}}^{f}$ also subsumes an interesting class of divergences for any submodular function representable as concave over modular. Consider any decomposable submodular function [24] $f$, representable as: $f(X) = \sum_i \lambda_i h_i(m_i(X))$, where the $h_i$s are (not necessarily smooth) concave functions and the $m_i$ are vectors in $\mathbb{R}^n$. Let $h_i'$ be any supergradient of $h_i$. Then we define $g_X^{cm} = \sum_i \lambda_i h_i'(m_i(X)) m_i$. Further we can define a divergence defined for a concave over modular function as:

$$d_{cm}^{f}(X,Y) = \sum_{i} \lambda_i (h_i(m_i(X)) - h_i(m_i(Y)) - h_i(m_i(X))(m_i(X) - m_i(Y)) \qquad (16)$$

Then it can be shown [12] that $d_{cm}^{f}$ is also a special case of $d_{\mathcal{G}^{f}}^{f}$ with $g_X = g_X^{cm}$ when $f$ is a decomposable submodular function.

Table 1: Instances of weighted distances measures as special cases of $d_f^{\mathcal{H}_f}$ and $d_{\mathcal{G}^f}^{f}$ for $w \in \mathbb{R}_+^n$

| Name | Type | $d$ | $f(X)$ | $\mathcal{H}_f(Y)/\mathcal{G}^f(X)$ |
|---|---|---|---|---|
| Hamming | $d_f^{\mathcal{H}_f}$ | $w(X\backslash Y) + w(Y\backslash X)$ | $w(X)$ | $2 \cdot w \odot 1_Y$ |
| Hamming | $d_{\mathcal{G}^f}^{f}$ | $w(X\backslash Y) + w(Y\backslash X)$ | $-w(X)$ | $-2 \cdot w \odot 1_X$ |
| Recall | $d_f^{\mathcal{H}_f}$ | $1 - \frac{w(X\cap Y)}{w(Y)}$ | $1$ | $\frac{w \odot 1_Y}{w(Y)}$ |
| Precision | $d_{\mathcal{G}^f}^{f}$ | $1 - \frac{w(X\cap Y)}{w(X)}$ | $-1$ | $-\frac{w \odot 1_X}{|X|}$ |
| AER$(Y,X;Y)$ | $d_f^{\mathcal{H}_f}$ | $1 - \frac{|Y|+|Y\cap X|}{2|Y|}$ | $\frac{1}{2}$ | $\frac{1_Y}{2|Y|}$ |
| Cond. MI | $d_f^{\sharp}$ | $I(\mathcal{X}_{X\backslash Y}; \mathcal{X}_{Y\backslash X}|\mathcal{X}_{X\cap Y})$ | $H(\mathcal{X}_X)$ | - |
| Itakura-Saito | $d_{\mathcal{G}^f}^{f}$ | $\frac{w(Y)}{w(X)} - \log\frac{w(Y)}{w(X)} - 1$ | $\log w(X)$ | $\frac{w}{w(X)}$ |
| Gen. KL | $d_{\mathcal{G}^f}^{f}$ | $w(Y)\log\frac{w(Y)}{w(X)} - w(Y) + w(X)$ | $-w(X)\log w(X)$ | $-w(1 + \log w(X))$ |

Finally both $d_f^{\mathcal{H}_f}$ and $d_{\mathcal{G}^f}^{f}$ generalize a number of interesting distance measures like Hamming, recall, precision, conditional mutual information, and weighted hamming. We show this in detail in [12], and owing to lack of space briefly summarize them in Table 1. The distance measures are shown in weighted form, but cardinality based distances are special cases with $w = \mathbf{1}$

## 2.3 The Lovász Bregman divergence

The Lovász extension ([20]) offers a natural connection between submodularity and convexity. The Lovász extension is a non-smooth convex function, and hence we can define a generalized Bregman divergence ([13, 18]) which has a number of properties and applications analogous to the Bregman divergence. Recall that the generalized Bregman divergence corresponding to a convex function $\phi$ is parameterized by the choice of the subgradient map $\mathcal{H}_\phi$. The Lovász extension of a submodular function has a very interesting set of subgradients, which have a particularly nice structure in that there is a very simple way of obtaining them [7].

Given a vector $y$, define a permutation $\sigma_y$ such that $y[\sigma_y(1)] \geq y[\sigma_y(2)] \geq \cdots \geq y[\sigma_y(n)]$ and define $Y_k = \{\sigma_y(1), \cdots, \sigma_y(k)\}$. The Lovász extension ([7, 20]) is defined as: $\hat{f}(y) = \sum_{k=1}^n y[\sigma_y(k)]f(\sigma_y(k)|Y_{k-1})$. For each point $y$, we can define a subdifferential $\partial \hat{f}(y)$, which has a particularly nice form [9]: for any point $y \in [0,1]^n$, $\partial \hat{f}(y) = \cap\{\partial_f(Y_i)|i = 1, 2 \cdots, n\}$. This naturally defines a generalized Bregman divergence $d_{\hat{f}}^{\mathcal{H}_{\hat{f}}}$ of the Lovász extension, parameterized by a subgradient map $\mathcal{H}_{\hat{f}}$, which we can define as:

$$d_{\hat{f}}^{\mathcal{H}_{\hat{f}}}(x,y) = \hat{f}(x) - \hat{f}(y) - \langle h_y, x-y\rangle, \text{ for some } h_y = H_{\hat{f}}(y) \in \partial \hat{f}(y). \qquad (17)$$

We can also define specific subgradients of $\hat{f}$ at $y$ as $h_{y,\sigma_y}$, with $h_{y,\sigma_y}(\sigma_y(k)) = f(Y_k) - f(Y_{k-1}), \forall k$. These subgradients are really the extreme points of the submodular polyhedron. Then define the Lovász Bregman divergence $d_{\hat{f}}$ as the Bregman divergence of $\hat{f}$ and the subgradient $h_{y,\sigma_y}$. Similar to $d_f^{\Sigma}$, it can be shown [12], that $d_{\hat{f}}(x,y) = \hat{f}(x) - \langle h_{y,\sigma_y}, x\rangle$. Note that if the vector $y$ is totally ordered (no two elements are equal to each other), the subgradient of $\hat{f}$ and the corresponding permutation $\sigma_y$ at $y$ will actually be unique. When the vector is not totally ordered, we can consider $\sigma_y$ as a permutation operator which defines a valid and consistent total ordering for every vector $y$, and we can then define the Bregman divergence in terms of it. Note also that the points with no total ordering in the interior of the hypercube is of measure zero. Hence for simplicity we just refer to the Lovász Bregman divergence as $d_{\hat{f}}$. The Lovász Bregman divergence is closely related to the lower bound submodular Bregman, as we show below.

**Theorem 2.2.** *The Lovász Bregman divergences are an extension of the lower bound submodular Bregman, over the interior of the hypercube. Further the Lovász Bregman divergence can be expressed as $d_{\hat{f}}(x,y) = \langle x, h_{x,\sigma_x} - h_{y,\sigma_y}\rangle$, and hence depends only $x$, the permutation $\sigma_x$ and the permutation of $y(\sigma_y)$, but is independent of the values of $y$.*

# 3 Properties of the submodular Bregman and Lovász Bregman divergences

In this section, we investigate some of the properties of the submodular Bregman and Lovász Bregman divergences which make these divergences interesting for Machine Learning applications. We only state them here — for an elaborate discussion refer to [12]. All forms of the submodular Bregman divergences are non-negative, and hence they are valid divergences. The lower bound submodular Bregman is submodular in $X$ for a given $Y$, while the upper bound submodular Bregman is supermodular in $Y$ for a given $X$. A direct consequence of this is that problems involving optimization in $X$ or $Y$ (for example in finding the discrete representatives in a discrete k-means like application which we consider in [12]), can be performed either exactly or approximately in polynomial time. In addition to these the forms of the submodular Bregman divergence also satisfy interesting properties like a characterization of equivalence classes, a form of set separation, a generalized triangle inequality over sets and a form of both Fenchel and submodular duality. Finally the generalized submodular Bregman divergence has an interesting alternate characterization, which shows that they can potentially subsume a large number of discrete divergences. In particular, a divergence $d$ is of the form $d_f^{\mathcal{H}_f}$ iff for any sets $A, B \subseteq V$, the set function $f_A(X) = d(X, A)$ is submodular in $X$ and the set function $d(X, A) - d(X, B)$ is modular in $X$. Similarly a divergence $d$ is of the form $d_{\mathcal{G}^f}^f$ iff, for any set $A, B \subseteq V$, the set function $f_A(Y) = d(A, Y)$ is supermodular in $Y$ and the set function $d(A, Y) - d(B, Y)$ is modular in $Y$. These facts show that the generalized Bregman divergences are potentially a very large class of divergences while Table 1 provides just a few of them.

Additionally, the Lovász Bregman divergence also has a number of very interesting properties. Notable amongst these is the fact that it has an interesting property related to permutations.

**Theorem 3.1.** *[12] Given a submodular function whose polyhedron contains all possible extreme points (e.g., $f(X) = \sqrt{|X|}$), $d_{\hat{f}}(x, y) = 0$ if and only if $\sigma_x = \sigma_y$.*

Hence the Lovász Bregman divergence can be seen as a divergence between the permutations. While a number of distance measures capture the notion of a distance amongst orderings [17], the Lovász Bregman divergences has a unique feature not present in these distance measures. The Lovász Bregman divergences not only capture the distance between $\sigma_x$ and $\sigma_y$, but also weighs it with the value of $x$, thus giving preference to the values and not just the orderings. Hence it can be seen as a divergence between a score $x$ and a permutation $\sigma_y$, and hence we shall also represent it as $d_{\hat{f}}(x, y) = d_{\hat{f}}(x||\sigma_y) = \langle x, h_{x,\sigma_x} - h_{x,\sigma_y} \rangle$. Correspondingly, given a collection of scores, it also measures how confident the scores are about the ordering. For example given two scores $x$ and $y$ with the same orderings such that the values of $x$ are nearly equal (low confidence), while the values of $y$ have large differences, the distance to any other permutation will be more for $y$ than $x$. This property intuitively desirable in a permutation based divergence. Finally, as we shall see the Lovász Bregman divergences are easily amenable to $k$-means style alternating minimization algorithms for clustering ranked data, a process that is typically difficult using other permutation-based distances.

# 4 Applications

In this section, we show the utility of the submodular Bregman and Lovász Bregman divergences by considering some practical applications in machine learning and optimization. The first application is that of proximal algorithms which generalize several mirror descent algorithms. As a second application, we motivate the use of the Lovász Bregman divergence as a natural choice in clustering where the order is important. Due to lack of space, we only concisely describe these applications, and for a more elaborate discussion please see [12] where we also consider a third discrete clustering application, and provide a clustering framework for the submodular Bregman with fast algorithms for clustering sets of binary vectors

## 4.1 A proximal framework for the submodular Bregman divergence

The Bregman divergence has some nice properties related to a proximal method. In particular ([5]), let $\psi$ be a convex function that is hard to optimize, but suppose the function $\psi(x) + \lambda d_\phi(x, y)$ is easy to

optimize for a given fixed $y$. Then a proximal algorithm, which starts with a particular $x^0$ and updates at every iteration $x^{t+1} = \text{argmax}_x \psi(x) + \lambda d_\phi(x, x^t)$, is bound to converge to the global minima.

We define a similar framework for the submodular Bregmans. Consider a set function $F$, and an underlying combinatorial constraint $\mathcal{S}$. Optimizing this set function may not be easy — e.g., if $\mathcal{S}$ is the constraint that $X$ be a graph-cut, then this optimization problem is NP hard even if $F$ is submodular ([15]). Consider

---
**Algorithm 1:** Proximal Minimization Algorithm

$X^0 = \emptyset$
**while** *until convergence* **do**
$\quad$ $X^{t+1} := \text{argmin}_{X \in \mathcal{S}} F(X) + \lambda d(X, X^t)$
$\quad$ $t \leftarrow t + 1$

---

now a divergence $d(X, Y)$ that can be either an upper or lower bound submodular Bregman. Note, the combinatorial constraints $\mathcal{S}$ are the discrete analogs of the convex set projection in the proximal method. We offer a proximal minimization algorithm (Algorithm 1) in a spirit similar to [5]. Furthermore, Algorithm 1 is guaranteed to monotonically decrease the function value over the iterations [12]. Interestingly, a number of approximate optimization problems considered in the past turn out to be special cases of the proximal framework. We show this below:

**Minimizing the difference between submodular (DS) functions:** Consider the case where $F(X) = f(X) - g(X)$ is a difference between two submodular functions $f$ and $g$. This problem is known to be NP hard and even NP hard to approximate [22, 11]. However there are a number of heuristic algorithms which have been shown to perform well in practice [22, 11]. Consider first: $d(X, X^t) = d_g^{\Sigma_t}(X, X^t)$ (for some appropriate schedule $\Sigma_t$ of permutations), with $\lambda = 1$ and $\mathcal{S} = 2^V$. Then it can be shown trivially [12] that we obtain the submodular-supermodular (sub-sup) procedure ([22]). Moreover, we can define $d(X, X^t) = d_{1:3}^f(X^t, X)$, again with $\lambda = 1$ and $\mathcal{S} = 2^V$. Then again we can show [12] that we obtain the supermodular-submodular (sup-sub) procedure [11]. Finally defining $d(X, X^t) = d_{1:3}^f(X^t, X) + d_g^{\Sigma_t}(X, X^t)$, we get the modular-modular (mod-mod) procedure [11]. Further, the sup-sub and mod-mod procedures can be used with more complicated constraints like cardinality, matroid and knapsack constraints while the mod-mod algorithm can be extended with even combinatorial constraints like the family of cuts, spanning trees, shortest paths, covers, matchings, etc. [11]

**Submodular function minimization:** Algorithm 1 also generalizes a number of approximate submodular minimization algorithms. If $F$ is a submodular function and the underlying constraints $\mathcal{S}$ represent the family of cuts, then we obtain the cooperative cut problem ([15], [14]) and one of the algorithms developed in ([15]) is a special case of Algorithm 1. If $\mathcal{S} = 2^V$ above, we get a form of the approximate submodular minimization algorithm suggested for arbitrary (non-graph representable) submodular functions ([16]). The proximal minimization algorithm also generalizes three submodular function minimization algorithms IMA-I, II and III, described in detail in [12] again with $\lambda = 1, \mathcal{S} = 2^V$ and $d = d_1^f, d_2^f$ and $d_3^f$ respectively. These algorithms are similar to the greedy algorithm for submodular maximization [23]. Interestingly these algorithms provide bounds to the lattice of minimizers of the submodular functions. It is known [1] that the sets $A = \{j : f(j|\emptyset) < 0\}, B = \{j : f(j|V - \{j\}) > 0\}$ are such that, for every minimizer $X^*$, $A \subseteq X^* \subseteq B$. Thus the lattice formed with $A$ and $B$ defined as the join and meet respectively, gives a bound on the minimizers, and we can restrict the submodular minimization algorithms to this lattice. However using $d = d_3^f$ as a regularizer (which is IMA-III) and starting with $X^0 = \emptyset$ and $X^0 = V$, we get the sets $A$ and $B$ [10, 12] respectively from Algorithm 1. With versions of algorithm 1 with $d = d_1^f$ and $d = d_2^f$, and starting respectively from $X^0 = \emptyset$ and $X^0 = V$, we get sets that provide a tighter bound on the lattice of minimizers than the one obtained with $A$ and $B$. Further these algorithms also provide improved bounds in the context of monotone submodular minimization subject to combinatorial constraints. In particular, these algorithms provide bounds which are better than $\frac{1}{\nu}$, where $\nu$ is a parameter related to the curvature of the submodular function. Hence when the parameter $\nu$ is a constant, these bounds are constant factor guarantees, which contrasts the $O(n)$ bounds for most of these problems. For a more elaborate and detailed discussion related to this, refer to [10]

**Submodular function maximization:** If $f$ is a submodular function, then using $d(X, X^v) = d_f^{\Sigma_v}(X, X^v)$ forms an iterative algorithm for maximizing the modular lower bound of a submodular function. This algorithm then generalizes a algorithms number of unconstrained submodular maximization and constrained submodular maximization, in that by an appropriate schedule of $\Sigma_v$ we can obtain these algorithms. Notable amongst them is a $\frac{1}{2}$ approximate algorithm and a $1 - \frac{1}{e}$

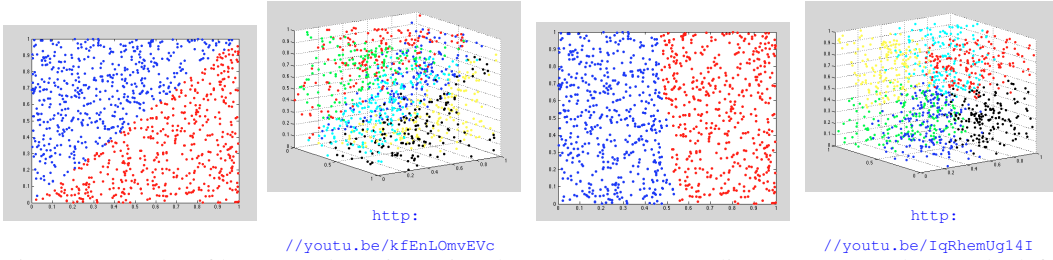

Figure 1: Results of k-means clustering using the Lovász Bregman divergence (two plots on the left) and the Euclidean distance (two plots on the right). URLs above link to videos.

approximation algorithm for unconstrained and cardinality constrained submodular maximization respectively. For a complete list of algorithms generalized by this, refer to [10].

## 4.2 Clustering framework with the Lovász Bregman divergence

In this section we investigate a clustering framework similar to [2], using the Lovász Bregman divergence and show how this is natural for a number of applications. Recall that the Lovász Bregman divergence in some sense measures the distance between the ordering of the vectors and can be seen as a form of the "sorting" distance. We define the clustering problem as given a set of vectors, find a clustering into subsets of vectors with similar orderings. For example, given a set of voters and their corresponding ranked preferences, we might want to find subsets of voters who mostly agree. Let $\mathcal{X} = \{x_1, x_2, \cdots, x_m\}$ represent a set of $m$ vectors, such that $\forall i, x_i \in [0,1]^n$. We first consider the problem of finding the representative of these vectors. Given a set of vectors $\mathcal{X}$ and a Lovász Bregman divergence $d_{\hat{f}}$, a natural choice of a representative (in this case a permutation) is the point with minimum average distance, or in other words: $\sigma = \operatorname{argmin}_{\sigma'} \sum_{i=1}^{n} d_{\hat{f}}(x_i || \sigma')$. Interestingly for the Lovász Bregman divergence this problem is easy and the representative permutation is exactly the permutation of the arithmetic mean of $\mathcal{X}$

**Theorem 4.1.** *[12] Given a submodular function $f$, the Lovász Bregman representative* $\operatorname{argmin}_{\sigma'} \sum_{i=1}^{n} d_{\hat{f}}(x_i || \sigma')$ *is exactly $\sigma = \sigma_\mu, \mu = \frac{1}{n} \sum_{i=1}^{n} x_i$*

It may not suffice to encode $\mathcal{X}$ using a single representative, and hence we partition $\mathcal{X}$ into disjoint blocks $\mathcal{C} = \{C_1, \cdots, C_k\}$ with each block having its own Lovász Bregman representative, with the set of representatives given by $\mathcal{M} = \{\sigma_1, \sigma_2, \cdots, \sigma_k\}$. Then we define an objective, which captures this idea of clustering vectors into subsets of similar orderings: $\min_{\mathcal{M}, \mathcal{C}} \sum_{j=1}^{k} \sum_{x_i \in C_j} d_{\hat{f}}(x_i, \mu_j)$. Consider then a $k$-means like alternating algorithm [19, 21]. It has two stages, often called the *assignment* and the *re-estimation* step. In the assignment stage, for every point $x_i$ we choose its cluster membership $C_j$ such that $j = \operatorname{argmin}_l d_{\hat{f}}(x_i || \sigma_l)$. The re-estimation step involves finding the representatives for every cluster $C_j$, which is exactly the permutation of the mean of the vectors in $C_j$. We skip the algorithm here due to space constraints, and refer the reader to [12] for a complete discussion.

We remark here that a number of distance measures capture the notion of orderings, like the bubble-sort distance [17], etc. However for these distance measures, finding the representative may not be easy. The Lovász Bregman divergence naturally captures the notion of distance between orderings of vectors and yet, the problem of finding the representative in this case is very easy. Finally similar to the analysis in [2, 12], we can show that the $k$-means algorithm using the Lovász Bregman divergence will monotonically decrease the objective at every iteration, and the algorithm converges to a local minima. [12] To demonstrate the utility of our clustering framework, we show some results in 2 and 3 dimensions (Fig. 1), where we compare our framework to a $k$-means algorithm using the euclidean distance. We use the submodular function $f(X) = \sqrt{w(X)}$, for an arbitrary vector $w$ ensuring unique base extreme points. The results clearly show that the Lovász Bregman divergence clusters the data based on the orderings of the vectors.

**Acknowledgments:** We thank Stefanie Jegelka, Karthik Narayanan, Andrew Guillory, Hui Lin, John Halloran and the rest of the submodular group at UW-EE for discussions. This material is based upon work supported by the National Science Foundation under Grant No. (IIS-1162606), and is also supported by a Google, a Microsoft, and an Intel research award.

# References

[1] F. Bach. Learning with Submodular functions: A convex Optimization Perspective. *Arxiv preprint*, 2011.

[2] A. Banerjee, S. Meregu, I. S. Dhilon, and J. Ghosh. Clustering with Bregman divergences. *JMLR*, 6:1705–1749, Oct. 2005.

[3] E. Boros and P. L. Hammer. Pseudo-boolean optimization. *Discrete Applied Math.*, 123(1–3):155 – 225, 2002.

[4] L. Bregman. The relaxation method of finding the common point of convex sets and its application to the solution of problems in convex programming. *USSR Comput. Math and Math Physics*, 7, 1967.

[5] Y. Censor and S. Zenios. *Parallel optimization: Theory, algorithms, and applications*. Oxford University Press, USA, 1997.

[6] I. Dhillon and J. Tropp. Matrix nearness problems with Bregman divergences. *SIAM Journal on Matrix Analysis and Applications*, 29(4):1120–1146, 2007.

[7] J. Edmonds. Submodular functions, matroids and certain polyhedra. *Combinatorial structures and their Applications*, 1970.

[8] B. Frigyik, S. Srivastava, and M. Gupta. Functional Bregman divergence. In *In Proc. ISIT*, pages 1681–1685. IEEE, 2008.

[9] S. Fujishige. *Submodular functions and optimization*, volume 58. Elsevier Science, 2005.

[10] R. Iyer and J. Bilmes. A framework of mirror descent algorithms for submodular optimization. *To Appear in NIPS Workshop on Discrete Optimization in Machine Learning (DISCML) 2012- Structure and Scalability*, 2012.

[11] R. Iyer and J. Bilmes. Algorithms for approximate minimization of the difference between submodular functions, with applications. *In Proc. UAI*, 2012.

[12] R. Iyer and J. Bilmes. Submodular-Bregman and the Lovász-Bregman Divergences with Applications: Extended Version. 2012.

[13] R. Iyer and J. Bilmes. A unified theory on the generalized bregman divergences. *Manuscript*, 2012.

[14] S. Jegelka and J. Bilmes. Cooperative cuts: Graph cuts with submodular edge weights. Technical report, Technical Report TR-189, Max Planck Institute for Biological Cybernetics, 2010.

[15] S. Jegelka and J. Bilmes. Submodularity beyond submodular energies: coupling edges in graph cuts. In *Computer Vision and Pattern Recognition (CVPR)*, Colorado Springs, CO, June 2011.

[16] S. Jegelka, H. Lin, and J. Bilmes. Fast approximate submodular minimization. *In Proc. NIPS*, 2011.

[17] M. Kendall. A new measure of rank correlation. *Biometrika*, 30(1/2):81–93, 1938.

[18] K. C. Kiwiel. Free-steering relaxation methods for problems with strictly convex costs and linear constraints. *Mathematics of Operations Research*, 22(2):326–349, 1997.

[19] S. Lloyd. Least squares quantization in pcm. *IEEE Transactions on IT*, 28(2):129–137, 1982.

[20] L. Lovász. Submodular functions and convexity. *Mathematical Programming*, 1983.

[21] J. MacQueen et al. Some methods for classification and analysis of multivariate observations. In *Proceedings of the fifth Berkeley symposium on math. stats and probability*, volume 1, pages 281–297. California, USA, 1967.

[22] M. Narasimhan and J. Bilmes. A submodular-supermodular procedure with applications to discriminative structure learning. In *Uncertainty in Artificial Intelligence (UAI)*, Edinburgh, Scotland, July 2005.

[23] G. Nemhauser, L. Wolsey, and M. Fisher. An analysis of approximations for maximizing submodular set functions—i. *Mathematical Programming*, 14(1):265–294, 1978.

[24] P. Stobbe and A. Krause. Efficient minimization of decomposable submodular functions. In *Proc. Neural Information Processing Systems (NIPS)*, 2010.

[25] M. Telgarsky and S. Dasgupta. Agglomerative bregman clustering. *In Proc. ICML*, 2012.

[26] K. Tsuda, G. Ratsch, and M. Warmuth. Matrix exponentiated gradient updates for on-line learning and Bregman projection. *JMLR*, 6(1):995, 2006.

[27] M. K. Warmuth. Online learning and Bregman divergences. *Tutorial at the Machine Learning Summer School*, 2006.

